# Sparseness of Support Vector Machines—Some Asymptotically Sharp Bounds

**Ingo Steinwart**
Modeling, Algorithms, and Informatics Group, CCS-3, Mail Stop B256
Los Alamos National Laboratory
Los Alamos, NM 87545, USA
ingo@lanl.gov

## Abstract

The decision functions constructed by support vector machines (SVM's) usually depend only on a subset of the training set—the so-called support vectors. We derive asymptotically sharp lower and upper bounds on the number of support vectors for several standard types of SVM's. In particular, we show for the Gaussian RBF kernel that the fraction of support vectors tends to twice the Bayes risk for the L1-SVM, to the probability of noise for the L2-SVM, and to 1 for the LS-SVM.

## 1 Introduction

Given a training set $T = ((x_1, y_1), \ldots, (x_n, y_n))$ with $x_i \in X$, $y_i \in Y := \{-1, 1\}$ standard support vector machines (SVM's) for classification (cf. [1], [2]) solve

$$\arg\min_{\substack{f \in H \\ b \in \mathbb{R}}} \lambda \|f\|_H^2 + \frac{1}{n} \sum_{i=1}^{n} L\big(y_i(f(x_i) + b)\big), \qquad (1)$$

where $H$ is a reproducing kernel Hilbert space (RKHS) of a kernel $k : X \times X \to \mathbb{R}$ (cf. [3], [4]), $\lambda > 0$ is a free regularization parameter and $L : \mathbb{R} \to [0, \infty)$ is a convex loss function.

Common choices for $L$ are the hinge loss function $L(t) := \max\{0, 1-t\}$, the squared hinge loss function $L(t) := (\max\{0, 1-t\})^2$ and the least square loss function $L(t) := (1-t)^2$. The corresponding classifiers are called L1-SVM, L2-SVM and LS-SVM, respectively.

Common choices of kernels are the Gaussian RBF $k(x, x') = \exp(-\sigma^2 \|x - x'\|_2^2)$ for $x, x' \in \mathbb{R}^d$ and fixed $\sigma > 0$ and polynomial kernels $k(x, x') = (\langle x, x' \rangle + c)^m$ for $x, x' \in \mathbb{R}^d$ and fixed $c \geq 0$, $m \in \mathbb{N}$.

If $(f_{T,\lambda}, b_{T,\lambda}) \in H \times \mathbb{R}$ denotes a solution of (1) we have

$$f_{T,\lambda} = \frac{1}{2\lambda} \sum_{i=1}^{n} y_i \alpha_i^* k(x_i, .) \qquad (2)$$

for suitable coefficients $\alpha_1^*, \ldots, \alpha_n^* \in \mathbb{R}$ (cf. [5]). Obviously, only the samples $x_i$ with $\alpha_i^* \neq 0$ have an impact on $f_{T,\lambda}$. These samples are called support vectors. The fewer support vectors $f_{T,\lambda}$ has the faster it can be evaluated. Moreover, it is well known that

the number of support vectors $\#SV(f_{T,\lambda})$ of the representation of $f_{T,\lambda}$ (cf. Section 3 for a brief discusssion) also has a large impact on the time needed to solve (1) using the dual problem. Therefore, it is of high interest to know how many support vectors one can expect for a given classification problem. In this work we address this question by establishing asymptotically lower and upper bounds on the number of support vectors for typical situations.

The rest of the paper is organized as follows: in Section 2 we introduce some technical notions and recall recent results in the direction of the paper. In Section 3 our results are presented and discussed, and finally, in Section 4 their proofs can be found.

## 2  Notations and known results

The standard assumption in classification is that the training set $T$ consists of i.i.d. pairs drawn from an unknown distribution $P$ on $X \times Y$. For technical reason we assume throughout this paper that $X$ is a compact metric space, e.g. a bounded, closed subset of $\mathbb{R}^d$. A Bayes decision function (cf. [6]) $f_P : X \to Y$ is a function that $P_X$-a.s. equals 1 and $-1$ on $C_1 := \{x \in X : P(1|x) > 1/2\}$ and $C_{-1} := \{x \in X : P(-1|x) > 1/2\}$, respectively. The corresponding classification error $\mathcal{R}_P$ of such a function is called the Bayes risk of $P$. Recall, that the Bayes risk is the smallest possible classification error.

A RKHS $H$ is called universal if $H$ is $\|.\|_\infty$-dense in the space of continuous functions $C(X)$. The best known example of a universal kernel is the Gaussian RBF kernel (cf. [7]).

Let us recall some results of the recent paper [8]. To simplify the statements, let us assume that $P$ has no discrete components, i.e. $P_X(\{x\}) = 0$ for all $x \in X$. Furthermore, let $L$ be a continuous convex loss function satisfying some minor regularity conditions. Then it was shown for universal RKHS's and stritly positive nullsequences $(\lambda_n)$ satisfying a regularity condition that the following statements hold for all $\varepsilon > 0$ and $n \to \infty$:

$$P^n\Big(T \in (X \times Y)^n : \#SV(f_{T,\lambda_n}) \geq (\mathcal{R}_P - \varepsilon)n\Big) \to 1\,. \tag{3}$$

In particular, this result holds for L1-SVM's. Furthermore, for $L$ being also differentiable (e.g. L2-SVM's and LS-SVM's) it was proved

$$P^n\Big(T \in (X \times Y)^n : \#SV(f_{T,\lambda_n}) \geq (\mathcal{S}_P - \varepsilon)n\Big) \to 1\,, \tag{4}$$

where $\mathcal{S}_P := P_X(\{x \in X : 0 < P(1|x) < 1\})$ denotes the probability of the set of points where noise occurs. Obviously, we always have $\mathcal{S}_P \geq 2\mathcal{R}_P$ and for *noisy non-degenerate* $P$, that is for $P$ with

$$P_X\big(\{x \in X : P(1|x) \notin \{0, 1/2, 1\}\}\big) > 0$$

this relation becomes a strict inequality. We shall prove in the next section that (3) can be significantly improved for the L1-SVM. We shall also show that this new lower bound is also an upper bound under moderate conditions on $P$ and $H$. Furthermore, we prove that (4) is asymptotically optimal for the L2-SVM and show that it can be significantly improved for the LS-SVM.

## 3  New bounds

We begin with lower and upper bounds for the L1-SVM. Recall, that the problem (1) for this classifier can be reformulated as

$$
\begin{array}{llll}
\text{minimize} & \lambda\langle f, f\rangle + \frac{1}{n}\sum_{i=1}^{n}\xi_i & \text{for } f \in H,\, b \in \mathbb{R},\, \xi \in \mathbb{R}^n & \\
\text{subject to} & y_i\big(f(x_i) + b\big) \geq 1 - \xi_i, & i = 1, \ldots, n & \\
& \xi_i \geq 0, & i = 1, \ldots, n\,. &
\end{array} \tag{5}
$$

Instead of solving (5) directly, one usually solves the dual optimization problem (cf. [4])

$$\text{maximize} \quad \sum_{i=1}^{n} \alpha_i - \frac{1}{4\lambda} \sum_{i,j=1}^{n} y_i y_j \alpha_i \alpha_j k(x_i, x_j) \qquad \text{for } \alpha \in \mathbb{R}^n$$

$$\text{subject to} \quad \sum_{i=1}^{n} y_i \alpha_i = 0, \tag{6}$$

$$0 \le \alpha_i \le \tfrac{1}{n}, \qquad\qquad\qquad i = 1, \ldots, n \,.$$

If $(\alpha_1^*, \ldots, \alpha_n^*) \in \mathbb{R}$ denotes a solution of (6) then $f_{T,\lambda}$ can be computed by (2). Note that the representaion of $f_{T,\lambda}$ is *not unique* in general, i.e. using other algorithms for solving (5) can lead to possibly sparser representations. However, in contrast to the general case the representation (2) of $f_{T,\lambda}$ is $P^n$-almost surely (a.s.) unique if the kernel is universal and $P$ has no discrete components (cf. [8]). Since our results for the L1-SVM hold for general kernels we always assume that $f_{T,\lambda}$ is found by (6). Finally, for a loss function $L$ and a RKHS $H$ we write

$$\mathcal{R}_{L,P,H} := \inf_{\substack{f \in H \\ b \in \mathbb{R}}} \mathcal{R}_{L,P}(f + b) \,,$$

where $\mathcal{R}_{L,P}(f) := \mathbb{E}_{(x,y) \sim P} L\big(yf(x)\big)$. Note, that $f_{T,\lambda_n} + b_{T,\lambda_n}$ cannot achieve an $L$-risk better than $\mathcal{R}_{L,P,H}$, if $H$ is the RKHS used in (1). Now, our first result is:

**Theorem 3.1** *Let $k$ be a continuous kernel on $X$ and $P$ be a probability measure on $X \times Y$ with no discrete components. Then for the L1-SVM using a regularization sequence $(\lambda_n)$ with $\lambda_n \to 0$ and $n\lambda_n^2 / \log n \to \infty$ and all $\varepsilon > 0$ we have*

$$P^n \Big( T \in (X \times Y)^n : \#SV(f_{T,\lambda_n}) \ge (\mathcal{R}_{L,P,H} - \varepsilon)n \Big) \to 1 \,.$$

**Remark 3.2** If $k$ is a universal kernel we have $\mathcal{R}_{L,P,H} = 2\mathcal{R}_P$ (cf. Ste7) and thus Theorem 3.1 yields the announced improvement of (3). For non-universal kernels we even have $\mathcal{R}_{L,P,H} > 2\mathcal{R}_P$ in general.

**Remark 3.3** For specific kernels the regularity condition $n\lambda_n^2 / \log n \to \infty$ can be weakened. Namely, for the Gaussian RBF kernel on $X \subset \mathbb{R}^d$ it can be substituted by $n\lambda_n |\log \lambda_n|^{-d-1} \to \infty$. Only slightly stronger conditions are sufficient for $C^\infty$-kernels. The interested reader can prove such conditions by establishing (9) using the results of [9].

**Remark 3.4** If $H$ is finite dimensional and $n > \dim H$ the representation (2) of $f_{T,\lambda_n}$ can be simplified such that only at most $\dim H$ kernel evaluations are neccessary. However, this simplification has no impact on the time needed for solving (6).

In order to formulate an upper bound on $\#SV(f_{T,\lambda_n})$ recall that a function is called analytic if it can be locally represented by its Taylor series. Let $L$ be a loss function, $H$ be a RKHS over $X$ and $P$ be a probability measure on $X \times Y$. We call the pair $(H, P)$ non-trivial (with respect to $L$) if

$$\mathcal{R}_{L,P,H} < \inf_{b \in \mathbb{R}} \mathcal{R}_{L,P}(b) \,,$$

i.e. the incorporation of $H$ has a non-trivial effect on the $L$-risk of $P$. If $H$ is universal we have $\mathcal{R}_{L,P,H} = \inf\{\mathcal{R}_{L,P}(f) \;\; f : X \to \mathbb{R}\}$ (cf. [9]) and therefore $(H, P)$ is non-trivial if $P$ has two non-vanishing classes, i.e. $P_X(C_1) > 0$ and $P_X(C_{-1}) > 0$. Furthermore, we denote the open unit ball of $\mathbb{R}^d$ by $B_{\mathbb{R}^d}$. Now our upper bound is:

**Theorem 3.5** *Let $H$ be the RKHS of an analytic kernel on $B_{\mathbb{R}^d}$. Furthermore, let $X \subset B_{\mathbb{R}^d}$ be a closed ball and $P$ be a noisy non-degenerate probability measure on $X \times Y$ such that*

$P_X$ *has a density with respect to the Lebesgue measure on $X$ and $(H, P)$ is non-trivial. Then for the L1-SVM using a regularization sequence $(\lambda_n)$ which tends sufficiently slowly to 0 we have*

$$\frac{\#SV(f_{T,\lambda_n})}{n} \to \mathcal{R}_{L,P,H}$$

*in probability.*

Probably the most restricting condition on $P$ in the above theorem is that $P_X$ has to have a density with respect to the Lebesgue measure. Considering the proof this condition can be slightly weakened to the assumption that every $d-1$-dimensional subset of $X$ has measure zero. Although it would be desirable to exclude only probability measures with discrete components it is almost obvious that such a condition cannot be sufficient for $d > 1$ (cf. [10, p.32]). The assumption that $P$ is noisy and non-degenerate is far more less restrictive since neither completely noise-free $P$ nor noisy problems with only "coin-flipping" noise often occur in practice. Finally, the condition that $(H, P)$ is non-trivial is more or less implicitly assumed whenever one uses nontrivial classifiers.

**Example 3.6** Theorem 3.5 directly applies to polynomial kernels. Note, that the limit $\mathcal{R}_{L,P,H}$ depends on both $P$ and the choice of the kernel.

**Example 3.7** Let $k$ be a Gaussian RBF kernel with RKHS $H$ and $X$ be a closed ball of $\mathbb{R}^d$. Moreover, let $P$ and $(\lambda_n)$ be according to Theorem 3.5. Recall, that $k$ is universal and hence $(H, P)$ is non-trivial iff $P$ has two non-vanishing classes. Since $k$ is also analytic on $\mathbb{R}^d$ we find

$$\frac{\#SV(f_{T,\lambda_n})}{n} \to 2\,\mathcal{R}_P\,.$$

Therefore, (4) shows that in general this L1-SVM produces sparser decision functions than the L2-SVM and the LS-SVM based on a Gaussian RBF kernel (cf. also Theorem 3.11).

**Remark 3.8** A variant of the L1-SVM that is often considered in theoretical papers is based on the optimization problem (5) with a-priori fixed $b := 0$. Besides the constraint $\sum_{i=1}^{n} y_i \alpha_i = 0$, which no longer appears, the corresponding dual problem is identical to (6). Hence it is easily seen that Theorem 3.1 also holds for this classifier. Moreover, for this modification Theorem 3.5 can be simplified. Namely, the assumption that $P$ is noisy and non-degenerate is superfluous (cf. [8, Prop. 33] to guarantee (14)). In particular, for a Gaussian RBF kernel and noise-free problems $P$ we then obtain

$$\frac{\#SV(f_{T,\lambda_n})}{n} \to 0\,, \tag{7}$$

i.e. the number of support vectors increases more slowly than linearly. This motivates the often claimed sparseness of SVM's.

The following theorem shows that the lower bound (4) on $\#SV(f_{T,\lambda_n})$ for the L2-SVM is often asymptotically optimal. This result is independent of the used optimization algorithm since we only consider universal kernels and measures with no discrete components.

**Theorem 3.9** *Let $H$ be the RKHS of an analytic and universal kernel on $B_{\mathbb{R}^d}$. Furthermore, let $X \subset B_{\mathbb{R}^d}$ be a closed ball and $P$ be a probability measure on $X \times Y$ with $\mathcal{R}_P > 0$ such that $P_X$ has a density with respect to the Lebesgue measure on $X$ and $(H, P)$ is non-trivial. Then for the L2-SVM using using a regularization sequence $(\lambda_n)$ which tends sufficiently slowly to 0 we have*

$$\frac{\#SV(f_{T,\lambda_n})}{n} \to \mathcal{S}_P$$

*in probability.*

**Remark 3.10** For the L2-SVM with fixed offset $b := 0$ the assumption $\mathcal{R}_P > 0$ in the above theorem is superfluous (cf. proof of Theorem 3.9 and proof of [8, Prop. 33]). In particular, for a Gaussian RBF kernel and noise-free problems $P$ we obtain (7), i.e. for noise-free problems this classifier also tends to produce sparse solutions in the sense of Remark 3.8.

Our last result shows that LS-SVM's often tend to use almost every sample as a support vector:

**Theorem 3.11** *Let $H$ be the RKHS of an analytic and universal kernel on $B_{\mathbb{R}^d}$. Furthermore, let $X \subset B_{\mathbb{R}^d}$ be a closed ball and $P$ be a probability measure on $X \times Y$ such that $P_X$ has a density with respect to the Lebesgue measure on $X$ and $(H, P)$ is non-trivial. Then for the LS-SVM using a regularization sequence $(\lambda_n)$ which tends sufficiently slowly to 0 we have*

$$\frac{\#SV(f_{T,\lambda_n})}{n} \to 1$$

*in probability.*

**Remark 3.12** Note, that unlike the L1-SVM and the L2-SVM (with fixed offset) the LS-SVM does not tend to produce sparse decision functions for noise-free $P$. This still holds if one fixes the offset for L2-SVM's, i.e. one considers regularization networks (cf. [11]). The reason for the different behaviours is the margin as already observed in [12]: the assumptions on $H$ and $P$ ensure that only a very small fraction of samples $x_i$ can be mapped to $\pm 1$ by $f_{T,\lambda_n}$ (cf. also Remark 4.1). For the L2-SVM this asymptotically ensures that most of the samples are mapped to values outside the margin, i.e. $y_i f_{T,\lambda_n}(x_i) > 1$, (cf. the properties of $B_n \setminus A_\delta$ in the proof of Theorem 3.9) and it is well-known that such samples cannot be support vectors. In contrast to this the LS-SVM has the property that every point not lying on the margin is a support vector. Using the techniques of our proofs it is fairly easy to see that the same reasoning holds for the hinge loss function compared to "modified hinge loss functions with no margin".

## 4 Proofs

Let $L$ be a loss function and $T$ be a training set. For a function $f : X \to \mathbb{R}$ we denote the empirical $L$-risk of $f$ by

$$\mathcal{R}_{L,T}(f + b) := \frac{1}{n} \sum_{i=1}^{n} L\big(y_i(f(x_i) + b)\big).$$

***Proof of Theorem 3.1:*** Let $(f_{T,\lambda_n}, b_{T,\lambda_n}, \xi^*) \in H \times \mathbb{R} \times \mathbb{R}^n$ and $\alpha^* \in \mathbb{R}^n$ be solutions of (5) and (6) for the regulariztion parameter $\lambda_n$, respectively. Since there is no duality gap between (5) and (6) we have (cf. [4]):

$$\lambda_n \langle f_{T,\lambda_n}, f_{T,\lambda_n} \rangle + \frac{1}{n} \sum_{i=1}^{n} \xi_i^* = \sum_{i=1}^{n} \alpha_i^* - \frac{1}{4\lambda_n} \sum_{i,j=1}^{n} y_i y_j \alpha_i^* \alpha_j^* k(x_i, x_j) \qquad (8)$$

By (2) this yields

$$\frac{1}{n} \sum_{i=1}^{n} \xi_i^* \leq 2\lambda_n \langle f_{T,\lambda_n}, f_{T,\lambda_n} \rangle + \frac{1}{n} \sum_{i=1}^{n} \xi_i^* = \sum_{i=1}^{n} \alpha_i^*.$$

Furthermore, recall that $\lambda_n \to 0$ and $n\lambda_n^2 / \log n \to \infty$ implies

$$\frac{1}{n} \sum_{i=1}^{n} \xi_i^* = \mathcal{R}_{L,T}(f_{T,\lambda_n} + b_{T,\lambda_n}) \to \mathcal{R}_{L,P,H} \qquad (9)$$

in probability for $n \to \infty$ (cf. [9]) and hence for all $\varepsilon > 0$ the probability of

$$\sum_{i=1}^{n} \alpha_i^* \geq \mathcal{R}_{L,P,H} - \varepsilon \qquad (10)$$

tends to 1 for $n \to \infty$. Now let us assume that our training set satisfies (10). Since $\alpha_i^* \leq 1/n$ we then find

$$\mathcal{R}_{L,P,H} - \varepsilon \leq \sum_{i=1}^{n} \alpha_i^* \leq \sum_{\alpha_i^* \neq 0} \frac{1}{n} = \frac{1}{n} \# SV(f_{T,\lambda_n})$$

which finishes the proof. ∎

For our further considerations we need to consider the optimization problem (1) with respect to $P$, i.e. we treat the (solvable, see [8]) problem

$$(f_{P,\lambda}, b_{P,\lambda}) := \arg\min_{\substack{f \in H \\ b \in \mathbb{R}}} \lambda \|f\|_H^2 + \mathcal{R}_{L,P}(f + b). \qquad (11)$$

***Proof of Theorem 3.5:*** Since $H$ is the RKHS of an analytic kernel every function $f \in H$ is analytic. Using the holomorphic extension of a non-constant $f \in H$ we see (after a suitable complex linear coordinate change, cf. [10, p. 31f]) that for $c \in \mathbb{R}$ and $x_1, \ldots, x_{d-1} \in \mathbb{R}$ the equation $f(x_1, \ldots, x_{d-1}, x_d) = c$ has at most $j$ solutions $x_d$, where $j \geq 0$ is locally (with respect to $x_1, \ldots, x_{d-1} \in \mathbb{R}$) constant . By a simple compactness argument we hence find

$$P_X\big(\{x \in X : f(x) = c\}\big) > 0 \qquad \Rightarrow \qquad f(x) = c \quad P_X\text{-a.s.} \qquad (12)$$

for all $f \in H$ and all $c \in \mathbb{R}$. Now, let us suppose that

$$P_X\big(\{x \in X : f_{P,\lambda}(x) + b_{P,\lambda} = f_P(x)\}\big) > 0 \qquad (13)$$

for some $\lambda > 0$, where $f_P$ denotes the Bayes decision function. Then we may assume without loss of generality that $P_X\big(\{x \in X : f_{P,\lambda}(x) + b_{P,\lambda} = 1\}\big) > 0$ holds. By (12) this leads to $f_{P,\lambda}(x) + b_{P,\lambda} = 1 \; P_X$-a.s. However, since $\mathcal{R}_{L,P}(f_{P,\lambda} + b_{P,\lambda}) \to \mathcal{R}_{L,P,H}$ for $\lambda \to 0$ (cf. [9]) we see that $f_{P,\lambda}$ cannot be constant for small $\lambda$ since $(H, P)$ was assumed to be non-trivial. Therefore (13) cannot hold for small $\lambda > 0$ and hence we may assume without loss of generality that

$$P_X\big(\{x \in X : |f_{P,\lambda}(x) + b_{P,\lambda} - f_P(x)| = 0\}\big) = 0$$

holds for all $\lambda > 0$. We define $A_\delta(\lambda) := \big\{x \in X : |f_{P,\lambda}(x) + b_{P,\lambda} - f_P(x)| \leq \delta\big\}$ for $\delta, \lambda > 0$. Our above considerations show that for all $\lambda > 0$ there exists a $\delta > 0$ with $P_X(A_\delta(\lambda)) \leq \varepsilon$. We write $\delta_\lambda := \frac{1}{2}\sup\{\delta > 0 : P_X(A_\delta(\lambda)) \leq \varepsilon\}$. We first show that there exists no sequence $\lambda_n \to \lambda \neq 0$ with $\delta_{\lambda_n} \to 0$. Let us assume the converse. Then there exists a subsequence with $(f_{P,\lambda_{n_j}}, b_{P,\lambda_{n_j}}) \to (f_{P,\lambda}, b_{P,\lambda})$ weakly and we have $\limsup_{j \to \infty} A_{3\delta_{\lambda_{n_j}}}(\lambda_{n_j}) \subset A_0(\lambda)$. By the construction we have $P_X(A_{3\delta_{\lambda_{n_j}}}(\lambda_{n_j})) \geq \varepsilon$ and hence $P_X(\limsup_{j \to \infty} A_{3\delta_{\lambda_{n_j}}}(\lambda_{n_j})) \geq \varepsilon$ by the Lemma of Fatou. This gives the contradiction $P_X(A_0(\lambda)) \geq \varepsilon$. Thus, the increasing function $\lambda \mapsto m(\lambda) := \inf\{\delta_{\tilde{\lambda}} : \tilde{\lambda} \geq \lambda\}$ satisfies $m(\lambda) > 0$ for all $\lambda > 0$. We fix a $T = ((x_1, y_1), \ldots, (x_n, y_n))$ with

$$\|f_{T,\lambda_n} + b_{T,\lambda_n} - f_{P,\lambda_n} - b_{P,\lambda_n}\|_\infty \leq \delta_n, \qquad (14)$$

$$\big|\mathcal{R}_{L,T}(f_{T,\lambda_n} + b_{T,\lambda_n}) - \mathcal{R}_{L,P}(f_{P,\lambda_n} + b_{P,\lambda_n})\big| \leq \varepsilon \qquad (15)$$

and $\big|\{i : x_i \in A_{\delta_n}(n))\}\big| \leq 2\varepsilon n$. If $m^4(\lambda_n)\lambda_n^3 n \to \infty$ the results of [9] and [8] ensure, that the probability of such a $T$ converges to 1 for $n \to \infty$. Moreover, by (8) we find

$$2\lambda_n \langle f_{T,\lambda_n}, f_{T,\lambda_n} \rangle + \mathcal{R}_{L,T}(f_{T,\lambda_n} + b_{T,\lambda_n}) = \sum_{i=1}^{n} \alpha_i^*. \qquad (16)$$

Since $f_{T,\lambda_n} + b_{T,\lambda_n}$ and $f_{P,\lambda_n} + b_{P,\lambda_n}$ minimize the regularized risks, (15) implies

$$\left| \lambda_n \|f_{T,\lambda_n}\|_H^2 + \mathcal{R}_{L,T}(f_{T,\lambda_n} + b_{T,\lambda_n}) - \lambda_n \|f_{P,\lambda_n}\|_H^2 - \mathcal{R}_{L,P}(f_{P,\lambda_n} + b_{P,\lambda_n}) \right| \leq \varepsilon. \quad (17)$$

Furthermore, if $n \to \infty$ we have

$$\lambda_n \|f_{P,\lambda_n}\|_H^2 + \mathcal{R}_{L,P}(f_{P,\lambda_n} + b_{P,\lambda_n}) \to \mathcal{R}_{L,P,H} \quad (18)$$

(cf. [9]) and therefore we obtain $\left| \lambda_n \|f_{T,\lambda_n}\|_H^2 + \mathcal{R}_{L,T}(f_{T,\lambda_n} + b_{T,\lambda_n}) - \mathcal{R}_{L,P,H} \right| \leq 2\varepsilon$ for large $n$. Now, (15), (17) and (18) implies $\lambda_n \langle f_{T,\lambda_n}, f_{T,\lambda_n} \rangle \leq 3\varepsilon$ for large $n$. Hence (16) yields

$$\mathcal{R}_{L,P,H} + 5\varepsilon \geq \sum_{i=1}^n \alpha_i^* \quad (19)$$

if $n$ is sufficiently large. Now let us suppose that we have a sample $(x_i, y_i)$ of $T$ with $x_i \notin A_{\delta_n}(n)$. Then we have $|f_{P,\lambda_n}(x_i) + b_{P,\lambda_n} - f_P(x_i)| > \delta_n$ and hence $f_{T,\lambda_n}(x_i) + b_{T,\lambda_n} \neq \pm 1$ by (14). By [4, p. 107] this means either $\alpha_i^* = 0$ or $\alpha_i^* = 1/n$. Therefore, by (19) we find

$$\mathcal{R}_{L,P,H} + 5\varepsilon \geq \sum_{i=1}^n \alpha_i^* \geq \sum_{x_i \notin A_{\delta_n}(n)} \alpha_i^* = \frac{1}{n} \left| \{ i : x_i \notin A_{\delta_n}(n) \text{ and } \alpha_i^* \neq 0 \} \right|$$

Since we have at most $2\varepsilon n$ samples in $A_{\delta_n}(n)$ we finally obtain

$$\frac{1}{n} \# SV(f_{T,\lambda_n}) \leq \mathcal{R}_{L,P,H} + 7\varepsilon.$$

Now the assertion follows by Theorem 3.1. ∎

**Remark 4.1** The proof of Theorem 3.5 is based on a kind of paradox: recall that it was shown in [8] that

$$f_{T,\lambda_n} + b_{T,\lambda_n} \to f_P$$

on $\{x \in X : P(1|x) \notin \{0, 1/2, 1\}\}$ in probability. However, the assumption on both $H$ and $P$ ensures that for typical $T$ the sets

$$\{ x \in X : |f_{T,\lambda_n}(x) + b_{T,\lambda_n} - f_P(x)| \leq \delta \}$$

become arbitrarily small for $\delta \to 0$. We will apply these seemingly contradicting properties in the following proofs, too.

***Proof of Theorem 3.9:*** Let $N := \{ x \in X : 0 < P(1|x) < 1 \}$ be the subset of $X$ where $P$ is noisy. Furthermore, let $A_\delta(n)$ be defined as in the proof of Theorem 3.5. We write

$$\begin{aligned} B_\delta(n) \; := \; & \left\{ x \in C_1 \setminus N : f_{P,\lambda_n}(x) + b_{P,\lambda_n} \geq 1 - \delta \right\} \\ & \cup \left\{ x \in C_{-1} \setminus N : f_{P,\lambda_n}(x) + b_{P,\lambda_n} \leq -1 + \delta \right\}. \end{aligned}$$

By [8, Thm. 22]) for all $n \geq 1$ there exists a $\delta > 0$ with $P_X(B_\delta(n)) \geq P_X(X \setminus N) - \varepsilon$. We define $\delta_n := \frac{1}{2} \sup\{ \delta > 0 : P_X(A_\delta(n)) \leq \varepsilon \text{ and } P_X(B_\delta(n)) \geq P_X(X \setminus N) - \varepsilon \}$. Let us fix a training set $T = ((x_1, y_1), \ldots, (x_n, y_n))$ with

$$\begin{aligned} \|f_{T,\lambda_n} + b_{T,\lambda_n} - f_{P,\lambda_n} - b_{P,\lambda_n}\|_\infty \; &\leq \; \delta_n, \\ \left| \{ i : x_i \in B_\delta(n) \setminus A_{\delta_n}(n) \} \right| \; &\geq \; n \left( P_X(X \setminus N) - 3\varepsilon \right). \end{aligned}$$

Again, the probability of such $T$ converges to 1 for $n \to \infty$ whenever $(\lambda_n)$ converges sufficiently slowly to 0. In view of (4) it suffices to show that every sample $x_i \in B_\delta(n) \setminus A_{\delta_n}(n)$ cannot be a support vector. Given an $x_i \in B_\delta(n) \setminus A_{\delta_n}(n)$ we may assume without loss of generality that $x_i \in C_1$. Then $x_i \in B_\delta(n)$ implies $f_{P,\lambda_n}(x_i) + b_{P,\lambda_n} \geq 1 - \delta_n$ while $x_i \notin A_{\delta_n}(n)$ yields $|f_{P,\lambda_n}(x_i) + b_{P,\lambda_n} - 1| > \delta_n$. Hence we find $f_{P,\lambda_n}(x_i) + b_{P,\lambda_n} > 1 + \delta_n$ and thus $f_{T,\lambda_n}(x_i) + b_{T,\lambda_n} > 1$. By the Karush-Kuhn-Tucker conditions of the primal/dual optimization problem of the L2-SVM (cf. [4, p. 105]) this shows that $x_i$ is not a support vector. ∎

***Proof of Theorem 3.11:*** Let $A_\delta(n)$ and $\delta_n$ be defined as in the proof of Theorem 3.5. Without loss of generality we may assume $\delta_n \in (0, 1/2)$. Let us define $C_0 := \{x \in X : P(1|x) = 1/2\}$ and

$$D_n = \left\{ x \in C_0 : |f_{P,\lambda_n}(x) + b_{P,\lambda_n}| \leq 1/2 \right\}.$$

By [8, Thm. 22] we may assume without loss of generality that $P_X(D_n) \geq P_X(C_0) - \varepsilon$ for all $n \geq 1$. Now, let us fix a training set $T = ((x_1, y_1), \ldots, (x_n, y_n))$ with

$$
\begin{aligned}
\|f_{T,\lambda_n} + b_{T,\lambda_n} - f_{P,\lambda_n} - b_{P,\lambda_n}\|_\infty &\leq \delta_n \\
\left|\{i : x_i \in A_{\delta_n}(n)\}\right| &\leq 2\varepsilon n \\
\left|\{i : x_i \in D_n\}\right| &\geq n\big(P_X(C_0) - 2\varepsilon\big).
\end{aligned}
$$

Again, the probability of such $T$ converges to 1 for $n \to \infty$ whenever $(\lambda_n)$ converges sufficiently slowly to 0. Now let us consider a sample $x_i \in (X \setminus A_{\delta_n}(n)) \cap C_1$ of $T$. Then we have $|f_{P,\lambda_n}(x_i) + b_{P,\lambda_n} - 1| > \delta_n$ and hence $f_{T,\lambda_n}(x_i) + b_{T,\lambda_n} \neq 1$. By [8, Cor. 32] this shows that $x_i$ is a support vector. Obviously, the same holds true for samples $x_i \in (X \setminus A_{\delta_n}(n)) \cap C_{-1}$. Finally, for samples $x_i \in D_n$ we have $|f_{T,\lambda_n}(x_i) + b_{T,\lambda_n}| \leq 1/2 + \delta_n < 1$ and hence these samples are always support vectors. ∎

## Acknowledgments

I would like to thank D. Hush and C. Scovel for helpful comments.

## References

[1] C. Cortes and V. Vapnik. Support vector networks. *Machine Learning*, 20:1995, 273–297.

[2] J.A.K. Suykens and J. Vandewalle. Least squares support vector machine classifiers. *Neural Processing Letters*, 9:293–300, 1999.

[3] N. Aronszajn. Theory of reproducing kernels. *Trans. Amer. Math. Soc.*, 68:337–404, 1950.

[4] N. Cristianini and J. Shawe-Taylor. *An Introduction to Support Vector Machines*. Cambridge University Press, 2000.

[5] B. Schölkopf, R. Herbrich, and A.J. Smola. A generalized representer theorem. In *Proceedings of the 14th Annual Conference on Computational Learning Theory*, pages 416–426. Lecture Notes in Artificial Intelligence 2111, 2001.

[6] L. Devroye, L. Györfi, and G. Lugosi. *A Probabilistic Theory of Pattern Recognition*. Springer, New York, 1997.

[7] I. Steinwart. On the influence of the kernel on the consistency of support vector machines. *Journal of Machine Learning Research*, 2:67–93, 2001.

[8] I. Steinwart. Sparseness of support vector machines. *Journal of Machine Learning Research*, 4:1071–1105, 2003.

[9] I. Steinwart. Consistency of support vector machines and other regularized kernel machine. *IEEE Transactions on Information Theory*, to appear.

[10] R.M. Range. *Holomorphic Functions and Integral Representations in Several Complex Variables*. Springer, 1986.

[11] F. Girosi, M. Jones, and T. Poggio. Regularization theory and neural networks architectures. *Neural Computation*, 7:219–269, 1995.

[12] A. Kowalczyk. Sparsity of data representation of optimal kernel machine and leave-one-out estimator. In T.K. Leen, T.G. Dietterich, and V. Tresp, editors, *Advances in Neural Information Processing Systems 13*, pages 252–258. MIT Press, 2001.
